# Catching Change-points with Lasso

**Zaid Harchaoui, Céline Lévy-Leduc**
LTCI, TELECOM ParisTech and CNRS
37/39 Rue Dareau, 75014 Paris, France
{zharchao,levyledu}@enst.fr

## Abstract

We propose a new approach for dealing with the estimation of the location of change-points in one-dimensional piecewise constant signals observed in white noise. Our approach consists in reframing this task in a variable selection context. We use a penalized least-squares criterion with a $\ell_1$-type penalty for this purpose. We prove some theoretical results on the estimated change-points and on the underlying piecewise constant estimated function. Then, we explain how to implement this method in practice by combining the LAR algorithm and a reduced version of the dynamic programming algorithm and we apply it to synthetic and real data.

## 1 Introduction

Change-points detection tasks are pervasive in various fields, ranging from audio [10] to EEG segmentation [5]. The goal is to partition a signal into several homogeneous segments of variable durations, in which some quantity remains approximately constant over time. This issue was addressed in a large literature (see [20] [11]), where the problem was tackled both from an online (sequential) [1] and an off-line (retrospective) [5] points of view. Most off-line approaches rely on a Dynamic Programming algorithm (DP), allowing to retrieve $K$ change-points within $n$ observations of a signal with a complexity of $\mathcal{O}(Kn^2)$ in time [11]. Such a feature refrains practitioners from applying these methods to large datasets. Moreover, one often observes a sub-optimal behavior of the raw DP algorithm on real datasets.

We suggest here to slightly depart from this line of research, by focusing on a reformulation of change-point estimation in a variable selection framework. Then, estimating change-point locations off-line turns into performing variable selection on dummy variables representing all possible change-point locations. This allows us to take advantage of the latest theoretical [23], [3] and practical [7] advances in regression with Lasso penalty. Indeed, Lasso provides us with a very efficient method for selecting potential change-point locations. This selection is then refined by using the DP algorithm to estimate the change-point locations.

Let us outline the paper. In Section 2, we first describe our theoretical reformulation of off-line change-point estimation as regression with a Lasso penalty. Then, we show that the estimated magnitude of jumps are close in mean, in a sense to be precized, to the true magnitude of jumps. We also give a non asymptotic inequality to upper-bound the $\ell_2$-loss of the true underlying piecewise constant function and the estimated one. We describe our algorithm in Section 3. In Section 4, we discuss related works. Finally, we provide experimental evidence of the relevance of our approach.

## 2 Theoretical approach

### 2.1 Framework

We describe, in this section, how off-line change-point estimation can be cast as a variable selection problem. Off-line estimation of change-point locations within a signal $(Y_t)$ consists in estimating the $\tau_k^\star$'s in the following model:

$$Y_t = \mu_k^\star + \varepsilon_t, \ \ t = 1, \ldots, n \ \text{ such that } \tau_{k-1}^\star + 1 \leq t \leq \tau_k^\star, \ 1 \leq k \leq K^\star \text{ with } \tau_0^\star = 0, \quad (1)$$

where $\varepsilon_t$ are i.i.d zero-mean random variables with finite variance. This problem can be reformulated as follows. Let us consider:

$$Y_n = X_n \beta^n + \varepsilon_n \tag{2}$$

where $Y_n$ is a $n \times 1$ vector of observations, $X_n$ is a $n \times n$ lower triangular matrix with nonzero elements equal to one and $\varepsilon_n = (\varepsilon_1^n, \ldots, \varepsilon_n^n)'$ is a zero-mean random vector such that the $\varepsilon_j^n$'s are i.i.d with finite variance. As for $\beta^n$, it is a $n \times 1$ vector having all its components equal to zero except those corresponding to the change-point instants. The above multiple change-point estimation problem (1) can thus be tackled as a variable selection one:

$$\underset{\beta}{\text{Minimize}} \ \|Y_n - X_n\beta\|_n^2 \text{ subject to } \|\beta\|_1 \leq s, \tag{3}$$

where $\|u\|_1$ and $\|u\|_n$ are defined for a vector $u = (u_1, \ldots, u_n) \in \mathbb{R}^n$ by $\|u\|_1 = \sum_{j=1}^n |u_j|$ and $\|u\|_n^2 = n^{-1} \sum_{j=1}^n u_j^2$ respectively. Indeed, the above formulation amounts to minimize the following counterpart objective in model (1):

$$\underset{\mu_1, \ldots, \mu_n}{\text{Minimize}} \ \frac{1}{n} \sum_{t=1}^n (Y_t - \mu_t)^2 \quad \text{subject to } \sum_{t=1}^{n-1} |\mu_{t+1} - \mu_t| \leq s, \tag{4}$$

which consists in imposing an $\ell_1$-constraint on the magnitude of jumps. The underpinning insight is the sparsity-enforcing property of the $\ell_1$-constraint, which is expected to give a sparse vector, whose non-zero components would match with those of $\beta^n$ and thus with change-point locations. It is related to the popular Least Absolute Shrinkage eStimatOr (LASSO) in least-square regression of [21], used for efficient variable selection.

In the next section, we provide two results supporting the use of the formulation (3) for off-line multiple change-point estimation. We show that estimates of jumps minimizing (3) are consistent in mean, and we provide a non asymptotic upper bound for the $\ell_2$ loss of the underlying estimated piecewise constant function and the true underlying piecewise function. This inequality shows that, at a precized rate, the estimated piecewise constant function tends to the true piecewise constant function with a probability tending to one.

### 2.2 Main results

In this section, we shall study the properties of the solutions of the problem (3) defined by

$$\hat{\beta}^n(\lambda) = \underset{\beta}{\text{Arg min}} \left\{ \|Y_n - X_n\beta\|_n^2 + \lambda\|\beta\|_1 \right\}. \tag{5}$$

Let us now introduce the notation sign. It maps positive entry to 1, negative entry to -1 and a null entry to zero. Let

$$\mathcal{A} = \{k, \ \beta_k^n \neq 0\} \text{ and } \overline{\mathcal{A}} = \{1, \ldots, n\} \backslash \mathcal{A} \tag{6}$$

and let $C^n$ the covariance matrix be defined by

$$C^n = n^{-1} X_n' X_n. \tag{7}$$

In a general regression framework, [18] recall that, with probability tending to one, $\hat{\beta}^n(\lambda)$ and $\beta^n$ have the same sign for a well-chosen $\lambda$, only if the following condition holds element-wise:

$$\left| C_{\overline{\mathcal{A}}\mathcal{A}}^n (C_{\mathcal{A}\mathcal{A}}^n)^{-1} \text{sign}(\beta_{\mathcal{A}}^n) \right| < 1, \tag{8}$$

where $C_{IJ}^n$ is a sub-matrix of $C^n$ obtained by keeping rows with index in the set $I$ and columns with index in $J$. The vector $\beta_{\mathcal{A}}^n$ is defined by $\beta_{\mathcal{A}}^n = (\beta_k^n)_{k \in \mathcal{A}}$. The condition (8) is not fulfilled in the

change-point framework implying that we cannot have a perfect estimation of the change-points as it is already known, see [13]. But, following [18] and [3], we can prove some consistency results, see Propositions 1 and 2 below.

In the following, we shall assume that the number of break points is equal to $K^\star$.

The following proposition ensures that for a large enough value of $n$ the estimated change-point locations are close to the true change-points.

**Proposition 1.** *Assume that the observations $(Y_n)$ are given by (2) and that the $\varepsilon_j^n$'s are centered. If $\lambda = \lambda_n$ is such that $\lambda_n\sqrt{n} \to 0$ as $n$ tends to infinity then*

$$\|\mathbb{E}(\hat{\beta}^n(\lambda_n)) - \beta^n\|_n \to 0 \ .$$

*Proof.* We shall follow the proof of Theorem 1 in [18]. For this, we denote $\beta^n(\lambda)$ the estimator $\hat{\beta}^n(\lambda)$ under the absence of noise and $\gamma_n(\lambda)$ the bias associated to the Lasso estimator: $\gamma_n(\lambda) = \beta^n(\lambda) - \beta^n$. For notational simplicity, we shall write $\gamma$ instead of $\gamma_n(\lambda)$. Note that $\gamma$ satisfies the following minimization: $\gamma = \text{Arg min}_{\zeta \in \mathbb{R}^n} f(\zeta)$ , where

$$f(\zeta) = \zeta' C^n \zeta + \lambda \sum_{k \in \mathcal{A}} |\beta_k^n + \zeta_k| + \lambda \sum_{k \in \bar{\mathcal{A}}} |\zeta_k| \ .$$

Since $f(\gamma) \leq f(0)$, we get

$$\gamma' C^n \gamma + \lambda \sum_{k \in \mathcal{A}} |\beta_k^n + \gamma_k| + \lambda \sum_{k \in \bar{\mathcal{A}}} |\gamma_k| \leq \lambda \sum_{k \in \mathcal{A}} |\beta_k^n| \ .$$

We thus obtain using the Cauchy-Schwarz inequality the following upper bound

$$\gamma' C^n \gamma \leq \lambda \sum_{k \in \mathcal{A}} |\gamma_k| \leq \lambda\sqrt{K^\star} \left( \sum_{k=1}^n |\gamma_k|^2 \right)^{1/2} \ .$$

Using that $\gamma' C^n \gamma \geq n^{-1} \sum_{k=1}^n |\gamma_k|^2$, we obtain: $\|\gamma\|_n \leq \lambda\sqrt{nK^\star}$. $\qquad \square$

The following proposition ensures, thanks to a non asymptotic result, that the estimated underlying piecewise function is close to the true piecewise constant function.

**Proposition 2.** *Assume that the observations $(Y_n)$ are given by (2) and that the $\varepsilon_j^n$'s are centered iid Gaussian random variables with variance $\sigma^2 > 0$. Assume also that $(\beta_k^n)_{k \in \mathcal{A}}$ belong to $(\beta_{min}, \beta_{max})$ where $\beta_{min} > 0$. For all $n \geq 1$ and $A > \sqrt{2}$ then, with a probability larger than $1 - n^{1-A^2/2}$, if $\lambda_n = A\sigma\sqrt{\log n/n}$,*

$$\|X_n(\hat{\beta}^n(\lambda_n) - \beta^n)\|_n^2 \leq 2A\sigma\beta_{max}K^\star\sqrt{\frac{\log n}{n}} \ .$$

*Proof.* By definition of $\hat{\beta}^n(\lambda)$ in (5) as a minimizer of a criterion, we have

$$\|Y_n - X_n\hat{\beta}^n(\lambda)\|_n^2 + \lambda\|\hat{\beta}^n(\lambda)\|_1 \leq \|Y_n - X_n\beta^n\|_n^2 + \lambda\|\beta^n\|_1 \ .$$

Using (2), we get

$$\|X_n(\beta^n - \hat{\beta}^n(\lambda))\|_n^2 + \frac{2}{n}(\beta^n - \hat{\beta}^n(\lambda))'X_n'\varepsilon_n + \lambda\sum_{j=1}^n |\hat{\beta}_j^n(\lambda)| \leq \lambda\sum_{j=1}^n |\beta_j^n| \ .$$

Thus,

$$\|X_n(\beta^n - \hat{\beta}^n(\lambda))\|_n^2 \leq \frac{2}{n}(\hat{\beta}^n(\lambda) - \beta^n)'X_n'\varepsilon_n + \lambda\sum_{j \in \mathcal{A}}(|\beta_j^n| - |\hat{\beta}_j^n(\lambda)|) - \lambda\sum_{j \in \bar{\mathcal{A}}} |\hat{\beta}_j^n(\lambda)| \ .$$

Observe that

$$\frac{2}{n}(\hat{\beta}^n(\lambda) - \beta^n)'X_n'\varepsilon_n = 2\sum_{j=1}^n (\hat{\beta}_j^n(\lambda) - \beta_j^n) \left( \frac{1}{n}\sum_{i=j}^n \varepsilon_i^n \right) \ .$$

Let us define the event $\mathcal{E} = \bigcap_{j=1}^{n} \left\{ n^{-1} \left| \sum_{i=j}^{n} \varepsilon_i^n \right| \leq \lambda \right\}$. Then, using the fact that the $\varepsilon_i^n$'s are iid zero-mean Gaussian random variables, we obtain

$$\mathbb{P}(\bar{\mathcal{E}}) \leq \sum_{j=1}^{n} \mathbb{P} \left( n^{-1} \left| \sum_{i=j}^{n} \varepsilon_i^n \right| > \lambda \right) \leq \sum_{j=1}^{n} \exp \left( -\frac{n^2 \lambda^2}{2\sigma^2 (n-j+1)} \right) .$$

Thus, if $\lambda = \lambda_n = A\sigma \sqrt{\log n / n}$,

$$\mathbb{P}(\bar{\mathcal{E}}) \leq n^{1-A^2/2} .$$

With a probability larger than $1 - n^{1-A^2/2}$, we get

$$\|X_n(\beta^n - \hat{\beta}^n(\lambda))\|_n^2 \leq \lambda_n \sum_{j=1}^{n} |\hat{\beta}_j^n(\lambda) - \beta_j^n| + \lambda_n \sum_{j \in \mathcal{A}} (|\beta_j^n| - |\hat{\beta}_j^n|) - \lambda_n \sum_{j \in \bar{\mathcal{A}}} |\hat{\beta}_j^n| .$$

We thus obtain with a probability larger than $1 - n^{1-A^2/2}$ the following upper bound

$$\|X_n(\beta^n - \hat{\beta}^n(\lambda))\|_n^2 \leq 2\lambda_n \sum_{j \in \mathcal{A}} |\beta_j^n| = 2A\sigma \sqrt{\frac{\log n}{n}} \sum_{j \in \mathcal{A}} |\beta_j^n| \leq 2A\sigma \beta_{max} K^\star \sqrt{\frac{\log n}{n}} .$$

$\square$

## 3 Practical approach

The previous results need to be efficiently implemented to cope with finite datasets. Our algorithm, called Cachalot (CAtching CHAnge-points with LassO), can be split into the following three steps described hereafter.

**Estimation with a Lasso penalty**  We compute the first $K_{\max}$ non-null coefficients $\hat{\beta}_{\tau_1}, \ldots, \hat{\beta}_{\tau_{K_{\max}}}$ on the regularization path of the LASSO problem (3). The LAR/LASSO algorithm, as described in [7], provides an efficient algorithm to compute the entire regularization path for the LASSO problem. Since $\sum_j |\beta_j| \leq s$ is a sparsity-enforcing constraint, the set $\{j, \ \hat{\beta}_j \neq 0\} = \{\tau_j\}$ becomes larger as we run through the regularization path. We shall denote by $S$ the $K_{\max}$-selected variables:

$$S = \left\{ \tau_1, \ldots, \tau_{K_{\max}} \right\} . \tag{9}$$

The computational complexity of the $K_{\max}$-long regularization path of LASSO solutions is $\mathcal{O}(K_{\max}^3 + K_{\max}^2 n)$. Most of the time, we can see that the Lasso effectively catches the true change-point but also irrelevant change-points at the vicinity of the true ones. Therefore, we propose to refine the set of change-points caught by the Lasso by performing a post-selection.

**Reduced Dynamic Programming algorithm**  One can consider several strategies to remove irrelevant change-points from the ones retrieved by the Lasso. Among them, since usually in applications, one is only interested in change-point estimation up to a given accuracy, we could launch the Lasso on a subsample of the signal. Here, we suggest to perform post-selection by using the standard Dynamic Programming algorithm (DP) thoroughly described in [11] (Chapter 12, p. 450) but on the reduced set $S$ instead of $\{1, \ldots, n\}$. This algorithm allows one to efficiently minimize the following objective for each $K$ in $\{1, \ldots, K_{\max}\}$:

$$J(K) = \underset{\substack{\tau_1 < \cdots < \tau_K \\ \text{s.t } \tau_1, \ldots, \tau_K \in S}}{\text{Min}} \sum_{k=1}^{K} \sum_{i=\tau_{k-1}+1}^{\tau_k} (Y_i - \hat{\mu}_k)^2, \tag{10}$$

$S$ being defined in (9) and outputs for each $K$, the corresponding subset of change-points $(\hat{\tau}_1, \ldots, \hat{\tau}_K)$. The DP algorithm has a computational complexity of $\mathcal{O}(K_{\max} n^2)$ if we look for at most $K_{\max}$ change-points within the signal. Here, our reduced DP calculations (rDP) scales as $\mathcal{O}(K_{\max} K_{\max}^2)$ where $K_{\max}$ is the maximum number of change-points/variables selected by LAR/LASSO algorithm. Since typically $K_{\max} \ll n$, our method thus provides a reduction of the computational burden associated with the classical change-points detection approach which consists in running the DP algorithm over all the $n$ observations.

**Selecting the number of change-points** The point is now to select the adequate number of change-points. As $n \to \infty$, according to [15], the ratio $\rho_k = J(k+1)/J(k)$ should show different qualitative behavior when $k \leqslant K^\star$ and when $k > K^\star$, $K^\star$ being the true number of change-points. In particular, $\rho_k \geq C_n$ for $k > K^\star$, where $C_n \to 1$ as $n \to \infty$. Actually we found out that $C_n$ was close to 1, even in small-sample settings, for various experimental designs in terms of noise variance and true number of change-points. Hence, conciliating theoretical guidance in large-sample setting and experimental findings in fixed-sample setting, we suggest the following rule of thumb for selecting the number of change-points $\hat{K}$ : $\hat{K} = \mathrm{Min}_{k \geq 1} \{\rho_k \geq 1 - \nu\}$, where $\rho_k = J(k+1)/J(k)$.

---

**Cachalot Algorithm**

**Input**

- Vector of observations $Y \in \mathbb{R}^n$
- Upper bound $K_{\max}$ on the number of change-points
- Model selection threshold $\nu$

**Processing**

1. Compute the first $K_{\max}$ non-null coefficients $(\beta_{\tau_1}, \ldots, \beta_{\tau_{K_{\max}}})$ on the regularization path with the LAR/LASSO algorithm.

2. Launch the rDP algorithm on the set of potential change-points $(\tau_1, \ldots, \tau_{K_{\max}})$.

3. Select the smallest subset of the potential change-points $(\tau_1, \ldots, \tau_{K_{\max}})$ selected by the rDP algorithm for which $\rho_k \geq 1 - \nu$.

**Output** Change-point locations estimates $\hat{\tau}_1, \ldots, \hat{\tau}_{\hat{K}}$.

---

To illustrate our algorithm, we consider observations $(Y_n)$ satisfying model (2) with $(\beta_{30}, \beta_{50}, \beta_{70}, \beta_{90}) = (5, -3, 4, -2)$, the other $\beta_j$ being equal to zero, $n = 100$ and $\varepsilon_n$ a Gaussian random vector with a covariance matrix equal to Id, Id being a $n \times n$ identity matrix. The set of the first nine active variables caught by the Lasso along the regularization path, *i.e.* the set $\{k, \hat{\beta}_k \neq 0\}$ is given in this case by: $S = \{21, 23, 28, 29, 30, 50, 69, 70, 90\}$. The set $S$ contains the true change-points but also irrelevant ones close to the true change-points. Moreover the most significant variables do not necessarily appear at the beginning. This supports the use of the reduced version of the DP algorithm hereafter. Table 1 gathers the $J(K), K = 1, \ldots, K_{\max}$ and the corresponding $(\hat{\tau}_1, \ldots, \hat{\tau}_K)$.

Table 1: Toy example: The empirical risk $J$ and the estimated change-points as a function of the possible number of change-points $K$

| K | J(K) | $(\hat{\tau}_1, \ldots, \hat{\tau}_K)$ |
|---|------|------------------|
| 0 | 696.28 | $\emptyset$ |
| 1 | 249.24 | 30 |
| 2 | 209.94 | (30,70) |
| 3 | 146.29 | (30,50,69) |
| 4 | 120.21 | (30,50,70,90) |
| 5 | 118.22 | (30,50,69,70,90) |
| 6 | 116.97 | (21,30,50,69,70,90) |
| 7 | 116.66 | (21,29,30,50,69,70,90) |
| 8 | 116.65 | (21,23,29,30,50,69,70,90) |
| 9 | 116.64 | (21,23,28,29,30,50,69,70,90) |

The different values of the ratio $\rho_k$ for $k = 0, \ldots, 8$ of the model selection procedure are given in Table 2. Here we took $\nu = 0.05$. We conclude, as expected, that $\hat{K} = 4$ and that the change-points are $(30, 50, 70, 90)$, thanks to the results obtained in Table 1.

## 4  Discussion

Off-line multiple change-point estimation has recently received much attention in theoretical works, both in a non-asymptotic and in an asymptotic setting by [17] and [13] respectively. From a practical point of view, retrieving the set of change-point locations $\{\tau_1^\star, \ldots, \tau_K^\star\}$ is challenging, since it is

Table 2: Toy example: The values of the ratio $(\rho_k = J(k+1)/J(k),\ k = 0,\dots,8)$

| $k$ | 0 | 1 | 2 | 3 | 4 | 5 | 6 | 7 | 8 |
|---|---|---|---|---|---|---|---|---|---|
| $\rho_k$ | 0.3580 | 0.8423 | 0.6968 | 0.8218 | 0.9834 | 0.9894 | 0.9974 | 0.9999 | 1.0000 |

plagued by the curse of dimensionality. Indeed, all of the $n$ observation times have to be considered as potential change-point instants. Yet, a dynamic programming algorithm (DP), proposed by [9] and [2], allows to explore all the configurations with a complexity of $\mathcal{O}(n^3)$ in time. Then selecting the number of change-points is usually performed thanks to a Schwarz-like penalty $\lambda_n K$, where $\lambda_n$ has to be calibrated on data [13] [12], or a penalty $K(a + b\log(n/K))$ as in [17] [14], where $a$ and $b$ are data-driven as well. We should also mention that an abundant literature tackles both change-point estimation and model selection issues from a Bayesian point of view (see [20] [8] and references therein). All approaches cited above rely on DP, or variants in Bayesian settings, and hence yield a computational complexity of $\mathcal{O}(n^3)$, which makes them inappropriate for very large-scale signal segmentation. Moreover, despite its theoretical optimality in a maximum likelihood framework, raw DP may sometimes have poor performances when applied to very noisy observations. Our alternative framework for multiple change-point estimation was previously elusively mentioned several times, e.g. in [16] [4] [19]. However up to our knowledge neither successful practical implementation nor theoretical grounding was given so far to support such an approach for change-point estimation. Let us also mention [22], where the Fused Lasso is applied in a similar yet different way to perform hot-spot detection. However, this approach includes an additional penalty, penalizing departures from the overall mean of the observations, and should thus rather be considered as an outlier detection method.

## 5 Comparison with other methods

### 5.1 Synthetic data

We propose to compare our algorithm with a recent method based on a penalized least-squares criterion studied by [12]. The main difficulty in such approaches is the choice of the constants appearing in the penalty. In [12], a very efficient approach to overcome this difficulty has been proposed: the choice of the constants is completely data-driven and has been implemented in a toolbox available online at `http://www.math.u-psud.fr/~lavielle/programs/index.html`.

In the following, we benchmark our algorithm: A together with the latter method: B. We shall use Recall and Precision as relevant performance measures to analyze the previous two algorithms. More precisely, the Recall corresponds to the ratio of change-points retrieved by a method with those really present in the data. As for the Precision, it corresponds to the number of change-points retrieved divided by the number of suggested change-points. We shall also estimate the probability of false alarm corresponding to the number of suggested change-points which are not present in the signal divided by the number of true change-points.

To compute the precision and the recall of methods A and B, we ran Monte-Carlo experiments. More precisely, we sampled 30 configurations of change-points for each real number of change-points $K^\star$ equal to 5, 10, 15 and 20 within a signal containing 500 observations. Change-points were at least distant of 10 observations. We sampled 30 configurations of levels from a Gaussian distribution. We used the following setting for the noise: for each configuration of change-points and levels, we synthesized a Gaussian white noise such that the standard deviation is set to a multiple of the minimum magnitude jump between two contiguous segments, i.e. $\sigma = m\ \mathrm{Min}_k(\mu^*_{k+1} - \mu^*_k)$, $\mu^\star_k$ being the level of the $k$th segment. The number of noise replications was set to 10.

As shown in Tables 3, 4 and 5 below, our method A yields competitive results compared to method B with $1 - \nu = 0.99$ and $K_{\max} = 50$. Performances in recall are comparable whereas method A provides better results than method B in terms of precision and false alarm rate.

### 5.2 Real data

In this section, we propose to apply our method previously described to real data which have already been analyzed by Bayesian methods: the well-log data which are described in [20] and [6] and

Table 3: Precision of methods A and B

|  | $K^\star = 5$ | | $K^\star = 10$ | | $K^\star = 15$ | | $K^\star = 20$ | |
|---|---|---|---|---|---|---|---|---|
| Method | A | B | A | B | A | B | A | B |
| $m = 0.1$ | 0.81±0.15 | 0.71±0.29 | 0.89±0.08 | 0.8±0.22 | 0.95±0.05 | 0.86±0.13 | 0.97±0.03 | 0.91±0.09 |
| $m = 0.5$ | 0.8±0.16 | 0.73±0.29 | 0.89±0.08 | 0.8±0.21 | 0.95±0.05 | 0.86±0.13 | 0.97±0.03 | 0.92±0.09 |
| $m = 1.0$ | 0.78±0.17 | 0.71±0.27 | 0.88±0.09 | 0.78±0.21 | 0.93±0.06 | 0.85±0.13 | 0.96±0.04 | 0.9±0.09 |
| $m = 1.5$ | 0.73±0.19 | 0.66±0.28 | 0.84±0.1 | 0.79±0.2 | 0.93±0.06 | 0.84±0.13 | 0.95±0.04 | 0.9±0.1 |

Table 4: Recall of methods A and B

|  | $K^\star = 5$ | | $K^\star = 10$ | | $K^\star = 15$ | | $K^\star = 20$ | |
|---|---|---|---|---|---|---|---|---|
| Method | A | B | A | B | A | B | A | B |
| $m = 0.1$ | 0.99±0.02 | 0.99±0.02 | 1±0 | 1±0 | 0.99±0 | 0.99±0 | 0.99±0 | 1±0 |
| $m = 0.5$ | 0.98±0.04 | 0.99±0.03 | 0.99±0.01 | 0.99±0.01 | 0.99±0.01 | 0.99±0.01 | 0.99±0.01 | 1±0 |
| $m = 1.0$ | 0.95±0.08 | 0.94±0.08 | 0.96±0.06 | 0.96±0.05 | 0.97±0.03 | 0.97±0.04 | 0.97±0.03 | 0.98±0.02 |
| $m = 1.5$ | 0.85±0.16 | 0.87±0.15 | 0.92±0.07 | 0.91±0.09 | 0.94±0.06 | 0.94±0.06 | 0.95±0.04 | 0.96±0.04 |

Table 5: False alarm rate of methods A and B

|  | $K^\star = 5$ | | $K^\star = 10$ | | $K^\star = 15$ | | $K^\star = 20$ | |
|---|---|---|---|---|---|---|---|---|
| Method | A | B | A | B | A | B | A | B |
| $m = 0.1$ | 0.13±0.03 | 0.23±0.2 | 0.24±0.03 | 0.33±0.19 | 0.34±0.02 | 0.42±0.13 | 0.44±0.02 | 0.51±0.12 |
| $m = 0.5$ | 0.13±0.03 | 0.22±0.2 | 0.23±0.03 | 0.32±0.18 | 0.33±0.02 | 0.41±0.13 | 0.44±0.02 | 0.5±0.11 |
| $m = 1.0$ | 0.13±0.03 | 0.21±0.18 | 0.23±0.03 | 0.32±0.18 | 0.33±0.02 | 0.4±0.13 | 0.43±0.03 | 0.5±0.12 |
| $m = 1.5$ | 0.13±0.03 | 0.21±0.2 | 0.23±0.03 | 0.29±0.16 | 0.31±0.03 | 0.4±0.15 | 0.42±0.03 | 0.48±0.11 |

displayed in Figure 1. They consist in nuclear magnetic response measurements expected to carry information about rock structure and especially its stratification.

One distinctive feature of these data is that they typically contain a non-negligible amount of outliers. The multiple change-point estimation method should then, either be used after a data cleaning step (median filtering [6]), or explicitly make heavy-tailed noise distribution assumption. We restricted ourselves to a median filtering pre-processing. The results given by our method applied to the well-log data processed with a median filter are displayed in Figure 1 for $K_{\max} = 200$ and $1 - \nu = 0.99$. The vertical lines locate the change-points. We can note that they are close to those found out by [6] (P. 206) who used Bayesian techniques to perform change-points detection.

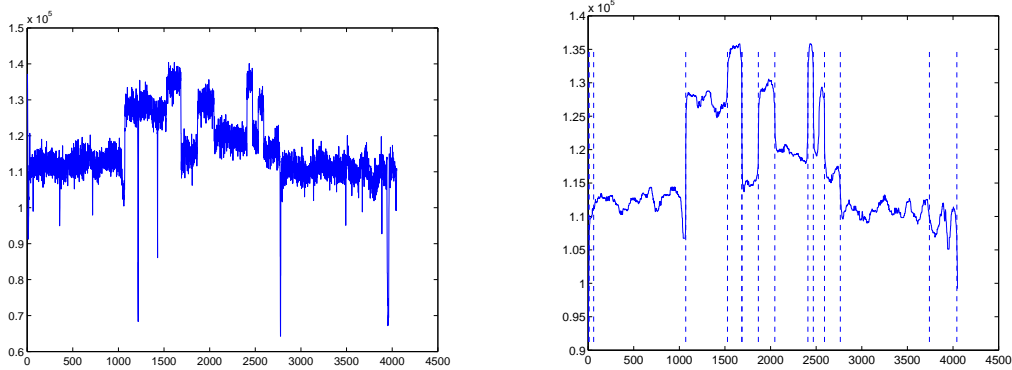

Figure 1: Left: Raw well-log data, Right: Change-points locations obtained with our method in well-log data processed with a median filter

# 6 Conclusion and prospects

We proposed here to cast the multiple change-point estimation as a variable selection problem. A least-square criterion with a Lasso-penalty yields an efficient primary estimation of change-point locations. Yet these change-point location estimates can be further refined thanks to a reduced dynamic programming algorithm. We obtained competitive performances on both artificial and real data, in terms of precision, recall and false alarm. Thus, Cachalot is a computationally efficient multiple change-point estimation method, paving the way for processing large datasets.

# References

[1] M. Basseville and N. Nikiforov. *The detection of abrupt changes*. Information and System sciences series. Prentice-Hall, 1993.

[2] R. Bellman. On the approximation of curves by line segments using dynamic programming. *Communications of the ACM*, 4(6), 1961.

[3] P. Bickel, Y. Ritov, and A. Tsybakov. Simultaneous analysis of Lasso and Dantzig selector. Preprint 2007.

[4] L. Boysen, A. Kempe, A. Munk, V. Liebscher, and O. Wittich. Consistencies and rates of convergence of jump penalized least squares estimators. *Annals of Statistics*, In revision.

[5] B. Brodsky and B. Darkhovsky. *Non-parametric statistical diagnosis: problems and methods*. Kluwer Academic Publishers, 2000.

[6] O. Cappé, E. Moulines, and T. Ryden. *Inference in Hidden Markov Models (Springer Series in Statistics)*. Springer-Verlag New York, Inc., 2005.

[7] B. Efron, T. Hastie, and R. Tibshirani. Least angle regression. *Annals of Statistics*, 32:407–499, 2004.

[8] P. Fearnhead. Exact and efficient bayesian inference for multiple changepoint problems. *Statistics and Computing*, 16:203–213, 2006.

[9] W. D. Fisher. On grouping for maximum homogeneity. *Journal of the American Statistical Society*, 53:789–798, 1958.

[10] O. Gillet, S. Essid, and G. Richard. On the correlation of automatic audio and visual segmentation of music videos. *IEEE Transactions on Circuits and Systems for Video Technology*, 2007.

[11] S. M. Kay. *Fundamentals of statistical signal processing: detection theory*. Prentice-Hall, Inc., 1993.

[12] M. Lavielle. Using penalized contrasts for the change-points problems. *Signal Processing*, 85(8):1501–1510, 2005.

[13] M. Lavielle and E. Moulines. Least-squares estimation of an unknown number of shifts in a time series. *Journal of time series analysis*, 21(1):33–59, 2000.

[14] E. Lebarbier. Detecting multiple change-points in the mean of a gaussian process by model selection. *Signal Processing*, 85(4):717–736, 2005.

[15] C.-B. L. Lee. Estimating the number of change-points in a sequence of independent random variables. *Statistics and Probability Letters*, 25:241–248, 1995.

[16] E. Mammen and S. Van De Geer. Locally adaptive regression splines. *Annals of Statistics*, 1997.

[17] P. Massart. A non asymptotic theory for model selection. pages 309–323. European Mathematical Society, 2005.

[18] N. Meinshausen and B. Yu. Lasso-type recovery of sparse representations for high-dimensional data. Preprint 2006.

[19] S. Rosset and J. Zhu. Piecewise linear regularized solution paths. *Annals of Statistics*, 35, 2007.

[20] J. Ruanaidh and W. Fitzgerald. *Numerical Bayesian Methods Applied to Signal Processing*. Statistics and Computing. Springer, 1996.

[21] R. Tibshirani. Regression shrinkage and selection via the lasso. *Journal of the Royal Statistical Society, Series B*, 58(1):267–288, 1996.

[22] R. Tibshirani and P. Wang. Spatial smoothing and hot spot detection for cgh data using the fused lasso. *Biostatistics*, 9(1):18–29, 2008.

[23] P. Zhao and B. Yu. On model selection consistency of lasso. *Journal Of Machine Learning Research*, 7, 2006.

